# Time Series Prediction Using Mixtures of Experts

**Assaf J. Zeevi**
Information Systems Lab
Department of Electrical Engineering
Stanford University
Stanford, CA. 94305
azeevi@isl.stanford.edu

**Ron Meir**
Department of Electrical Engineering
Technion
Haifa 32000, Israel
rmeir@ee.technion.ac.il

**Robert J. Adler**
Department of Statistics
University of North Carolina
Chapel Hill, NC. 27599
adler@stat.unc.edu

## Abstract

We consider the problem of prediction of stationary time series, using the architecture known as mixtures of experts (MEM). Here we suggest a mixture which blends several autoregressive models. This study focuses on some theoretical foundations of the prediction problem in this context. More precisely, it is demonstrated that this model is a *universal approximator*, with respect to learning the unknown prediction function. This statement is strengthened as upper bounds on the mean squared error are established. Based on these results it is possible to compare the MEM to other families of models (e.g., neural networks and state dependent models). It is shown that a degenerate version of the MEM is in fact *equivalent* to a neural network, and the number of experts in the architecture plays a similar role to the number of hidden units in the latter model.

# 1  Introduction

In this work we pursue a new family of models for time series, substantially extending, but strongly related to and based on the classic linear autoregressive moving average (ARMA) family. We wish to exploit the linear autoregressive technique in a manner that will enable a substantial increase in modeling power, in a framework which is non-linear and yet mathematically tractable.

The novel model, whose main building blocks are linear AR models, deviates from linearity in the integration process, that is, the way these blocks are combined. This model was first formulated in the context of a regression problem, and an extension to a hierarchical structure was also given [2]. It was termed the mixture of experts model (MEM).

Variants of this model have recently been used in prediction problems both in economics and engineering. Recently, some theoretical aspects of the MEM , in the context of non-linear regression, were studied by Zeevi *et al.* [8], and an equivalence to a class of neural network models has been noted.

The purpose of this paper is to extend the previous work regarding the MEM in the context of regression, to the problem of prediction of time series. We shall demonstrate that the MEM is a universal approximator, and establish upper bounds on the *approximation error*, as well as the mean squared error, in the setting of *estimation* of the predictor function.

It is shown that the MEM is intimately related to several existing, state of the art, statistical non-linear models encompassing Tong's TAR (threshold autoregressive) model [7], and a certain version of Priestley's [6] state dependent models (SDM). In addition, it is demonstrated that the MEM is equivalent (in a sense that will be made precise) to the class of feedforward, sigmoidal, neural networks.

# 2  Model Description

The MEM [2] is an architecture composed of $n$ *expert networks*, each being an $AR(d)$ linear model. The experts are combined via a *gating* network, which partitions the input space accordingly. Considering a scalar time series $\{x_t\}$, we associate with each expert a probabilistic model (density function) relating input vectors $\mathbf{x}_{t-d}^{t-1} \equiv [x_{t-1}, x_{t-2}, \ldots, x_{t-d}]$ to an output scalar $x_t \in \mathbb{R}$ and denote these probabilistic models by $p(x_t| \mathbf{x}_{t-d}^{t-1}; \boldsymbol{\theta}_j, \sigma_j)$ $j = 1, 2, \ldots, n$ where $(\boldsymbol{\theta}_j, \sigma_j)$ is the expert parameter vector, taking values in a compact subset of $\mathbb{R}^{d+1}$. In what follows we will use upper case $X_t$ to denote random variables, and lower case $x_t$ to denote values taken by those r.v.'s.

Letting the parameters of each expert network be denoted by $(\boldsymbol{\theta}_j, \sigma_j)$, $j = 1, 2, \ldots, n$, those of the gating network by $\boldsymbol{\theta}_g$ and letting $\Theta = (\{\boldsymbol{\theta}_j, \sigma_j\}_{j=1}^n, \boldsymbol{\theta}_g)$ represent the complete set of parameters specifying the model, we may express the conditional distribution of the model, $p(x_t|\mathbf{x}_{t-d}^{t-1}, \Theta)$, as

$$p(x_t|\mathbf{x}_{t-d}^{t-1}; \Theta) = \sum_{j=1}^n g_j(\mathbf{x}_{t-d}^{t-1}; \boldsymbol{\theta}_g)p(x_t|\mathbf{x}_{t-d}^{t-1}; \boldsymbol{\theta}_j, \sigma_j), \tag{1}$$

together with the constraint that $\sum_{j=1}^n g_j(\mathbf{x}_{t-d}^{t-1}; \boldsymbol{\theta}_g) = 1$ and $g_j(\mathbf{x}_{t-d}^{t-1}; \boldsymbol{\theta}_g) \geq$

$0 \quad \forall \mathbf{x}_{t-d}^{t-1}$. We assume that the parameter vector $\Theta \in \Omega$, a compact subset of $\mathbb{R}^{2n(d+1)}$.

Following the work of Jordan and Jacobs [2] we take the probability density functions to be Gaussian with mean $\boldsymbol{\theta}_j^T \mathbf{x}_{t-d}^{t-1} + \theta_{j,0}$ and variance $\sigma_j$ (representative of the underlying, *local* AR($d$) model). The function $g_j(\mathbf{x}; \boldsymbol{\theta}_g) \equiv \exp\{\boldsymbol{\theta}_{g_j}^T \mathbf{x} + \theta_{g_j,0}\}/(\sum_{i=1}^{n} \exp\{\boldsymbol{\theta}_{g_i}^T \mathbf{x} + \theta_{g_i,0}\}$, thus implementing a multiple output logistic regression function.

The underlying non-linear mapping (i.e., the conditional expectation, or $L_2$ prediction function) characterizing the MEM, is described by using (1) to obtain the conditional expectation of $X_t$,

$$f_n^\theta = \mathrm{E}[X_t | X_{t-d}^{t-1}; \mathcal{M}_n] = \sum_{j=1}^{n} g_j(X_{t-d}^{t-1}; \boldsymbol{\theta}_g)[\boldsymbol{\theta}_j^T X_{t-d}^{t-1} + \theta_{j,0}], \qquad (2)$$

where $\mathcal{M}_n$ denotes the MEM model. Here the subscript $n$ stands for the number of experts. Thus, we have $\hat{X}_t = f_n^\theta \equiv f_n(X_{t-d}^{t-1}; \Theta)$ where $f_n : \mathbb{R}^d \times \Omega \to \mathbb{R}$, and $\hat{X}_t$ denotes the projection of $X_t$ on the 'relevant past', given the model, thus defining the *model predictor function*.

We will use the notation MEM($n; d$) where $n$ is the number of experts in the model (proportional to the complexity, or number of parameters in the model), and $d$ the lag size. In this work we assume that $d$ is known and given.

## 3 Main results

### 3.1 Background

We consider a stationary time series, more precisely a discrete time stochastic process $\{X_t\}$ which is assumed to be strictly stationary. We define the $L_2$ predictor function

$$f \equiv \mathrm{E}[X_t | X_{-\infty}^{t-1}] = \mathrm{E}[X_t | X_{t-d}^{t-1}] \quad a.s.$$

for some fixed lag size $d$. Markov chains are perhaps the most widely encountered class of probability models exhibiting this dependence. The NAR($d$), that is non-linear AR($d$), model is another example, widely studied in the context of time series (see [4] for details). Assuming additive noise, the NAR($d$) model may be expressed as

$$X_t = f(X_{t-1}, X_{t-2}, \ldots, X_{t-d}) + \varepsilon_t . \qquad (3)$$

We note that in this formulation $\{\varepsilon_t\}$ plays the role of the *innovation* process for $X_t$, and the function $f(\cdot)$ describes the information on $X_t$ contained within its past history.

In what follows, we restrict the discussion to stochastic processes satisfying certain constraints on the memory decay, more precisely we are assuming that $\{X_t\}$ is an exponentially $\alpha$-mixing process. Loosely stated, this assumption enables the process to have a law of large numbers associated with it, as well as a certain version of the central limit theorem. These results are the basis for analyzing the asymptotic behavior of certain parameter estimators (see, [9] for further details), but other than that this assumption is merely stated here for the sake of completeness. We note in

passing that this assumption may be substantially weakened, and still allow similar results to hold, but requires more background and notation to be introduced, and therefore is not pursued in what follows (the reader is referred to [1] for further details).

## 3.2 Objectives

Knowing the $L_2$ predictor function, $f$, allows optimal prediction of future samples, where optimal is meant in the sense that the predicted value is the closest to the true value of the next sample point, in the mean squared error sense. It therefore seems a reasonable strategy, to try and *learn* the optimal predictor function, based on some finite realization of the stochastic process, which we will denote $\mathcal{D}_N = \{X_t\}_{t=0}^{d+N+1}$. Note that for $N \gg d$, the number of sample points is approximately $N$.

We therefore define our objective as follows. Based on the data $\mathcal{D}_N$, we seek the 'best' approximation to $f$, the $L_2$ predictor function, using the MEM$(n, d)$ predictor $f_n^\theta \in \mathcal{M}_n$ as the approximator model.

More precisely, define the least squares (LS) parameter estimator for the MEM$(n, d)$ as

$$\hat{\boldsymbol{\theta}}_{n,N} = \arg\min_{\theta \in \Theta} \sum_{t=d+1}^{N} \left[ X_t - f_n(X_{t-d}^{t-1}, \boldsymbol{\theta}) \right]^2$$

where $f_n(X_{t-d}^{t-1}, \boldsymbol{\theta})$ is $f_n^\theta$ evaluated at the point $X_{t-d}^{t-1}$, and define the LS functional estimator as

$$\hat{f}_{n,N} \equiv f_n^\theta|_{\theta = \hat{\theta}_{n,N}}$$

where $\hat{\boldsymbol{\theta}}_{n,N}$ is the LS parameter estimator.

Now, define the functional estimator risk as

$$\text{MSE}[f, \hat{f}_{n,N}] \equiv \text{E}_{\mathcal{D}} \left[ \int |f - \hat{f}_{n,N}|^2 d\nu \right]$$

where $\nu$ is the $d$ fold probability measure of the process $\{X_t\}$. In this work we maintain that the integration is over some compact domain $I^d \subset \mathbb{R}^d$, though recent work [3] has shown that the results can be extended to $\mathbb{R}^d$, at the price of slightly slower convergence rates.

It is reasonable, and quite customary, to expect a 'good' estimator to be one that is asymptotically unbiased. However, growth of the sample size itself need not, and in general does not, mean that the estimator is 'becoming' unbiased. Consequently, as a figure of merit, we may restrict attention to the *approximation capacity* of the model. That is we ask, what is the error in approximating a given class of predictor functions, using the MEM$(n, d)$ (i.e., $\{\mathcal{M}_n\}$) as the approximator class.

To measure this figure, we define the *optimal risk* as

$$\text{MSE}[f, f_n^\star] \equiv \int |f - f_n^\star|^2 d\nu,$$

where $f_n^\star \equiv f_n^\theta|_{\theta = \theta^\star}$ and

$$\theta_n^\star = \arg\min_{\theta \in \Theta} \int |f - f_n^\theta|^2 d\nu,$$

that is, $\theta_n^*$ is the parameter minimizing the expected $L_2$ loss function. One may think of $f_n^*$ as the 'best' predictor function in the class of approximators, i.e., the closest approximation to the optimal predictor, given the finite complexity, $n$, (i.e., finite number of parameters) of the model. Here $n$ is simply the number of experts (AR models) in the architecture.

## 3.3  Upper Bounds on the Mean Squared Error and Universal Approximation Results

Consider first the case where we are simply interested in approximating the function $f$, assuming it belongs to some class of functions. The question then arises as to how well one may approximate $f$ by a MEM architecture comprising $n$ experts. The answer to this question is given in the following proposition, the proof of which can be found in [8].

**Proposition 3.1** (Optimal risk bounds) *Consider the class of functions $\mathcal{M}_n$ defined in (2) and assume that the optimal predictor $f$ belongs to a Sobolev class containing $r$ continuous derivatives in $L_2$. Then the following bound holds:*

$$\text{MSE}[f, f_n^*] \leq \frac{c}{n^{2r/d}} \tag{4}$$

*where $c$ is a constant independent of $n$.*

PROOF SKETCH: The proof proceeds by first approximating the normalized gating function $g_j()$ by polynomials of finite degree, and then using the fact that polynomials can approximate functions in Sobolev space to within a known degree of approximation.

The following main theorem, establishing upper bounds on the functional estimator risk, constitutes the main result of this paper. The proof is given in [9].

**Theorem 3.1** (Upper bounds on the estimator risk)
*Suppose the stochastic process obeys the conditions set forth in the previous section. Assume also that the optimal predictor function, $f$, possesses $r$ smooth derivatives in $L_2$. Then for $N$ sufficiently large we have*

$$\text{MSE}[f, \hat{f}_{n,N}] \leq \frac{c}{n^{2r/d}} + \frac{m_n^*}{2N} + o\left(\frac{1}{N}\right) , \tag{5}$$

*where $r$ is the number of continuous derivatives in $L_2$ that $f$ is assumed to possess, $d$ is the lag size, and $N$ is the size of the data set $\mathcal{D}_N$.*

PROOF SKETCH: The proof proceeds by a standard stochastic Taylor expansion of the loss around the point $\theta_n^*$. Making common regularity assumptions [1] and using the assumption on the $\alpha$-mixing nature of the process allows one to establish the usual asymptotic normality results, from which the result follows.

We use the notation $m_n^*$ to denote the *effective number of parameters*. More precisely, $m_n^* = \text{Tr}\{B_n^*(A_n^*)^{-1}\}$ and the matrices $A^*$ and $B^*$ are related to the Fisher information matrix in the case of misspecified estimation (see [1] for further discussion). The upper bound presented in Theorem 3.1 is related to the classic *bias - variance* decomposition in statistics and the obvious tradeoffs are evident by inspection.

## 3.4   Comments

It follows from Proposition 3.1 that the class of mixtures of experts is a universal approximator, w.r.t. the class of target functions defined for the optimal predictor. Moreover, Proposition 3.1 establishes the rate of convergence of the approximator, and therefore relates the approximation error to the number of experts used in the architecture ($n$).

Theorem 3.1 enhances this result, as it relates the sample complexity and model complexity, for this class of models. The upper bounds may be used in defining *model selection* criteria, based on upper bound minimization. In this setting, we may use an estimator of the stochastic error bound (i.e., the estimation error), to penalize the complexity of the model, in the spirit of AIC, MDL etc (see [8] for further discussion).

At a first glance it may seem surprising to find that a combination of linear models is a universal function approximator. However, one must keep in mind that the *global* model is nonlinear, due to the gating network. Nevertheless, this result does imply, at least on a theoretical ground, that one may restrict the MEM($n, d$) to be locally linear, without loss of generality. Thus, taking a simple local model, enabling efficient and tractable learning algorithms (see [2]), still results in a rich global model.

## 3.5   Comparison

Recently, Mhaskar [5] proved upper bounds on a feedforward sigmoidal neural network, for target functions in the same class as we consider herein, i.e., the Sobolev class. The bound we have obtained in Proposition 3.1, and its extension in [8], demonstrate that w.r.t. to this particular target class, neural networks and mixtures of experts are equivalent. That is, both models attain optimal precision in the degree of approximation results (see [5] for details of this argument). Keeping in mind the advantages of the MEM with respect to learning and generalization [2], we believe that our results lend further credence to the emerging view as to the superiority of modular architectures over the more standard feedforward neural networks.

Moreover, the detailed proof of Proposition 3.1 (see [8]) actually takes the MEM($n, d$) to be made up of local *constants*. That is, the linear experts are degenerated to constant functions. Thus, one may conjecture that mixtures of experts are in fact a more general class than feedforward neural networks, though we have no proof of this as of yet.

Two nonlinear alternatives, generalizing standard statistical linear models, have been pointed out in the introductory section. These are Tong's TAR (threshold autoregressive) model [7], and the more general SDM (state dependent models) introduced by Priestley. The latter models can be reduced to a TAR model by imposing a more restrictive structure (for further details see [6]) . We have shown, based on the results described above (see [9]), that the MEM may be viewed as a generalization of the SDM (and consequently of the TAR model). The relation to the state dependent models is of particular interest, as the mixtures of experts is structured on state dependence as well. Exact statement and proofs of these facts can be found in [9].

We should also note that we have conducted several numerical experiments, comparing the performance of the MEM with other approaches. We tested the model on both synthetic as well as real-world data. Without any fine-tuning of parameters we found the performance of the MEM, with linear expert functions, to compare very favorably with other approaches (such as TAR , ARMA and neural networks). Details of the numerical results may be found in [9]. Moreover, the model also provided a very natural and intuitive segmentation of the process.

## 4 Discussion

In this work we have pursued a novel non-linear model for prediction in stationary time series. The mixture of experts model (MEM) has been demonstrated to be a rich model, endowed with a sound theoretical basis, and compares favorably with other, state of the art, nonlinear models.

We hope that the results of this study will aid in establishing the MEM as, yet another, powerful tool for the study of time-series applicable to the fields of statistics, economics, and signal processing.

## References

[1] Domowitz, I. and White, H. "Misspecified Models with Dependent Observations", *Journal of Econometrics*, vol. 20: 35-58, 1982.

[2] Jordan, M. and Jacobs, R. "Hierarchical Mixtures of Experts and the EM Algorithm", *Neural Computation*, vol. 6, pp. 181-214, 1994.

[3] Maiorov, V. and Meir. V. "Approximation Bounds for Smooth Functions in $C(\mathbb{R}^d)$ by Neural and Mixture Networks", submitted for publication, December 1996.

[4] Meyn, S.P. and Tweedie, R.L. (1993) *Markov Chains and Stochastic Stability*, Springer-Verlag, London.

[5] Mhaskar, H. (1996) "Neural Networks for Optimal Approximation of Smooth and Analytic Functions", *Neural Computation* vol. 8(1), pp. 164-177.

[6] Priestley M.B. *Non-linear and Non-stationary Time Series Analysis*, Academic Press, New York, 1988.

[7] Tong, H. *Threshold Models in Non-linear Time Series Analysis*, Springer Verlag, New York, 1983.

[8] Zeevi, A.J., Meir, R. and Maiorov, V. "Error Bounds for Functional Approximation and Estimation Using Mixtures of Experts", EE Pub. CC-132., Electrical Engineerin g Department, Technion, 1995.

[9] Zeevi, A.J., Meir, R. and Adler, R.J. "Non-linear Models for Time Series Using Mixtures of Experts", EE Pub. CC-150, Electrical Engineering Department, Technion, 1996.
